# Maximum-Margin Matrix Factorization

**Nathan Srebro**
Dept. of Computer Science
University of Toronto
Toronto, ON, CANADA
nati@cs.toronto.edu

**Jason D. M. Rennie**     **Tommi S. Jaakkola**
Computer Science and Artificial Intelligence Lab
Massachusetts Institute of Technology
Cambridge, MA, USA
jrennie,tommi@csail.mit.edu

## Abstract

We present a novel approach to collaborative prediction, using low-norm instead of low-rank factorizations. The approach is inspired by, and has strong connections to, large-margin linear discrimination. We show how to learn low-norm factorizations by solving a semi-definite program, and discuss generalization error bounds for them.

## 1 Introduction

Fitting a target matrix $Y$ with a low-rank matrix $X$ by minimizing the sum-squared error is a common approach to modeling tabulated data, and can be done explicitly in terms of the singular value decomposition of $Y$. It is often desirable, though, to minimize a different loss function: loss corresponding to a specific probabilistic model (where $X$ are the mean parameters, as in pLSA [1], or the natural parameters [2]); or loss functions such as hinge loss appropriate for binary or discrete ordinal data. Loss functions other than squared-error yield non-convex optimization problems with multiple local minima. Even with a squared-error loss, when only some of the entries in $Y$ are observed, as is the case for collaborative filtering, local minima arise and SVD techniques are no longer applicable [3].

Low-rank approximations constrain the dimensionality of the factorization $X = UV'$. Other constraints, such as sparsity and non-negativity [4], have also been suggested for better capturing the structure in $Y$, and also lead to non-convex optimization problems.

In this paper we suggest regularizing the factorization by constraining the norm of $U$ and $V$—constraints that arise naturally when matrix factorizations are viewed as feature learning for large-margin linear prediction (Section 2). Unlike low-rank factorizations, such constraints lead to *convex* optimization problems that can be formulated as semi-definite programs (Section 4). Throughout the paper, we focus on using low-norm factorizations for "collaborative prediction": predicting unobserved entries of a target matrix $Y$, based on a subset $S$ of observed entries $Y_S$. In Section 5, we present generalization error bounds for collaborative prediction using low-norm factorizations.

## 2 Matrix Factorization as Feature Learning

Using a low-rank model for collaborative prediction [5, 6, 3] is straightforward: A low-rank matrix $X$ is sought that minimizes a loss versus the observed entries $Y_S$. Unobserved

entries in $Y$ are predicted according to $X$. Matrices of rank at most $k$ are those that can be factored into $X = UV'$, $U \in \mathbb{R}^{n \times k}$, $V \in \mathbb{R}^{m \times k}$, and so seeking a low-rank matrix is equivalent to seeking a low-dimensional factorization.

If one of the matrices, say $U$, is fixed, and only the other matrix $V'$ needs to be learned, then fitting each column of the target matrix $Y$ is a separate linear prediction problem. Each row of $U$ functions as a "feature vector", and each column of $V'$ is a linear predictor, predicting the entries in the corresponding column of $Y$ based on the "features" in $U$.

In collaborative prediction, both $U$ and $V$ are unknown and need to be estimated. This can be thought of as learning feature vectors (rows in $U$) for each of the rows of $Y$, enabling good linear prediction across all of the prediction problems (columns of $Y$) concurrently, each with a different linear predictor (columns of $V'$). The features are learned without any external information or constraints which is impossible for a single prediction task (we would use the labels as features). The underlying assumption that enables us to do this in a collaborative filtering situation is that the prediction tasks (columns of $Y$) are *related*, in that the same features can be used for all of them, though possibly in different ways.

Low-rank collaborative prediction corresponds to regularizing by limiting the dimensionality of the feature space—each column is a linear prediction problem in a low-dimensional space. Instead, we suggest allowing an unbounded dimensionality for the feature space, and regularizing by requiring a low-norm factorization, while predicting with large-margin.

Consider adding to the loss a penalty term which is the sum of squares of entries in $U$ and $V$, i.e. $\|U\|_{\text{Fro}}^2 + \|V\|_{\text{Fro}}^2$ ($\|\|_{\text{Fro}}$ denotes the Frobenius norm). Each "conditional" problem (fitting $U$ given $V$ and vice versa) again decomposes into a collection of standard, this time regularized, linear prediction problems. With an appropriate loss function, or constraints on the observed entries, these correspond to large-margin linear discrimination problems. For example, if we learn a binary observation matrix by minimizing a hinge loss plus such a regularization term, each conditional problem decomposes into a collection of SVMs.

## 3  Maximum-Margin Matrix Factorizations

Matrices with a factorization $X = UV'$, where $U$ and $V$ have low Frobenius norm (recall that the dimensionality of $U$ and $V$ is no longer bounded!), can be characterized in several equivalent ways, and are known as low *trace norm* matrices:

**Definition 1.** *The trace norm[1] $\|X\|_\Sigma$ is the sum of the singular values of $X$.*

**Lemma 1.** $\|X\|_\Sigma = \min_{X=UV'} \|U\|_{Fro} \|V\|_{Fro} = \min_{X=UV'} \frac{1}{2}(\|U\|_{Fro}^2 + \|V\|_{Fro}^2)$

The characterization in terms of the singular value decomposition allows us to characterize low trace norm matrices as the convex hull of bounded-norm rank-one matrices:

**Lemma 2.** $\left\{ X \mid \|X\|_\Sigma \leq B \right\} = \text{conv} \left\{ uv' \mid u \in \mathbb{R}^n, v \in \mathbb{R}^m, |u|^2 = |v|^2 = B \right\}$

In particular, the trace norm is a convex function, and the set of bounded trace norm matrices is a convex set. For convex loss functions, seeking a bounded trace norm matrix minimizing the loss versus some target matrix is a convex optimization problem.

This contrasts sharply with minimizing loss over low-rank matrices—a non-convex problem. Although the sum-squared error versus a *fully observed* target matrix can be minimized efficiently using the SVD (despite the optimization problem being non-convex!), minimizing other loss functions, or even minimizing a squared loss versus a partially observed matrix, is a difficult optimization problem with multiple local minima [3].

In fact, the trace norm has been suggested as a convex surrogate to the rank for various rank-minimization problems [7]. Here, we justify the trace norm directly, both as a natural extension of large-margin methods and by providing generalization error bounds.

To simplify presentation, we focus on binary labels, $Y \in \{\pm 1\}^{n \times m}$. We consider *hard-margin matrix factorization*, where we seek a minimum trace norm matrix $X$ that matches the observed labels with a margin of one: $Y_{ia}X_{ia} \geq 1$ for all $ia \in S$. We also consider *soft-margin* learning, where we minimize a trade-off between the trace norm of $X$ and its hinge-loss relative to $Y_S$:

$$\text{minimize } \|X\|_\Sigma + c \sum_{ia \in S} \max(0, 1 - Y_{ia}X_{ia}). \tag{1}$$

As in maximum-margin linear discrimination, there is an inverse dependence between the norm and the margin. Fixing the margin and minimizing the trace norm is equivalent to fixing the trace norm and maximizing the margin. As in large-margin discrimination with certain infinite dimensional (e.g. radial) kernels, the data is always separable with sufficiently high trace norm (a trace norm of $\sqrt{n|S|}$ is sufficient to attain a margin of one).

**The max-norm variant** Instead of constraining the norms of rows in $U$ and $V$ on average, we can constrain all rows of $U$ and $V$ to have small $L_2$ norm, replacing the trace norm with $\|X\|_{\max} = \min_{X=UV'}(\max_i |U_i|)(\max_a |V_a|)$ where $U_i, V_a$ are rows of $U, V$. Low-max-norm discrimination has a clean geometric interpretation. First, note that predicting the target matrix with the signs of a rank-$k$ matrix corresponds to mapping the "items" (columns) to points in $\mathbb{R}^k$, and the "users" (rows) to homogeneous hyperplanes, such that each user's hyperplane separates his positive items from his negative items. Hard-margin low-max-norm prediction corresponds to mapping the users and items to points and hyperplanes in a high-dimensional unit sphere such that each user's hyperplane separates his positive and negative items with a large-margin (the margin being the inverse of the max-norm).

## 4 Learning Maximum-Margin Matrix Factorizations

In this section we investigate the optimization problem of learning a MMMF, i.e. a low norm factorization $UV'$, given a binary target matrix. Bounding the trace norm of $UV'$ by $\frac{1}{2}(\|U\|_{\text{Fro}}^2 + \|V\|_{\text{Fro}}^2)$, we can characterize the trace norm in terms of the trace of a positive semi-definite matrix:

**Lemma 3 ([7, Lemma 1]).** *For any $X \in \mathbb{R}^{n \times m}$ and $t \in \mathbb{R}$: $\|X\|_\Sigma \leq t$ iff there exists $A \in \mathbb{R}^{n \times n}$ and $B \in \mathbb{R}^{m \times m}$ such that [2] $\left[\begin{smallmatrix} A & X \\ X' & B \end{smallmatrix}\right] \succeq 0$ and $\operatorname{tr} A + \operatorname{tr} B \leq 2t$.*

*Proof.* Note that for any matrix $W$, $\|W\|_{\text{Fro}} = \operatorname{tr} WW'$. If $\left[\begin{smallmatrix} A & X \\ X' & B \end{smallmatrix}\right] \succeq 0$, we can write it as a product $\left[\begin{smallmatrix} U \\ V \end{smallmatrix}\right]\left[\begin{smallmatrix} U' & V' \end{smallmatrix}\right]$. We have $X = UV'$ and $\frac{1}{2}(\|U\|_{\text{Fro}}^2 + \|V\|_{\text{Fro}}^2) = \frac{1}{2}(\operatorname{tr} A + \operatorname{tr} B) \leq t$, establishing $\|X\|_\Sigma \leq t$. Conversely, if $\|X\|_\Sigma \leq t$ we can write it as $X = UV'$ with $\operatorname{tr} UU' + \operatorname{tr} VV' \leq 2t$ and consider the p.s.d. matrix $\left[\begin{smallmatrix} UU' & X \\ X' & VV' \end{smallmatrix}\right]$. $\qquad\square$

Lemma 3 can be used in order to formulate minimizing the trace norm as a semi-definite optimization problem (SDP). Soft-margin matrix factorization (1), can be written as:

$$\min \frac{1}{2}(\operatorname{tr} A + \operatorname{tr} B) + c \sum_{ia \in S} \xi_{ia} \text{ s.t. } \begin{bmatrix} A & X \\ X' & B \end{bmatrix} \succeq 0, \quad \begin{matrix} y_{ia}X_{ia} \geq 1 - \xi_{ia} \\ \xi_{ia} \geq 0 \end{matrix} \quad \forall ia \in S \tag{2}$$

Associating a dual variable $Q_{ia}$ with each constraint on $X_{ia}$, the dual of (2) is [8, Section 5.4.2]:

$$\max \sum_{ia \in S} Q_{ia} \quad \text{s.t.} \quad \begin{bmatrix} I & (-Q \otimes Y) \\ (-Q \otimes Y)' & I \end{bmatrix} \succcurlyeq 0, \quad 0 \le Q_{ia} \le c \qquad (3)$$

where $Q \otimes Y$ denotes the sparse matrix $(Q \otimes Y)_{ia} = Q_{ia} Y_{ia}$ for $ia \in S$ and zeros elsewhere. The problem is strictly feasible, and there is no duality gap. The p.s.d. constraint in the dual (3) is equivalent to bounding the spectral norm of $Q \otimes Y$, and the dual can also be written as an optimization problem subject to a bound on the spectral norm, i.e. a bound on the singular values of $Q \otimes Y$:

$$\max \sum_{ia \in S} Q_{ia} \quad \text{s.t.} \quad \begin{array}{l} \|Q \otimes Y\|_2 \le 1 \\ 0 \le Q_{ia} \le c \quad \forall ia \in S \end{array} \qquad (4)$$

In typical collaborative prediction problems, we observe only a small fraction of the entries in a large target matrix. Such a situation translates to a sparse dual semi-definite program, with the number of variables equal to the number of observed entries. Large-scale SDP solvers can take advantage of such sparsity.

The prediction matrix $X^*$ minimizing (1) is part of the primal optimal solution of (2), and can be extracted from it directly. Nevertheless, it is interesting to study how the optimal prediction matrix $X^*$ can be directly recovered from a dual optimal solution $Q^*$ alone. Although unnecessary when relying on interior point methods used by most SDP solvers (as these return a primal/dual optimal pair), this can enable us to use specialized optimization methods, taking advantage of the simple structure of the dual.

**Recovering $X^*$ from $Q^*$** As for linear programming, recovering a primal optimal solution directly from a dual optimal solution is not always possible for SDPs. However, at least for the hard-margin problem (no slack) this is possible, and we describe below how an optimal prediction matrix $X^*$ can be recovered from a dual optimal solution $Q^*$ by calculating a singular value decomposition and solving linear equations.

Given a dual optimal $Q^*$, consider its singular value decomposition $Q^* \otimes Y = U \Lambda V'$. Recall that all singular values of $Q^* \otimes Y$ are bounded by one, and consider only the columns $\tilde{U} \in \mathbb{R}^{n \times p}$ of $U$ and $\tilde{V} \in \mathbb{R}^{m \times p}$ of $V$ with singular value one. It is possible to show [8, Section 5.4.3], using complimentary slackness, that for some matrix $R \in \mathbb{R}^{p \times p}$, $X^* = \tilde{U} R R' \tilde{V}'$ is an optimal solution to the maximum margin matrix factorization problem (1). Furthermore, $\frac{p(p+1)}{2}$ is bounded above by the number of non-zero $Q_{ia}^*$. When $Q_{ia}^* > 0$, and assuming hard-margin constraints, i.e. no box constraints in the dual, complimentary slackness dictates that $X_{ia}^* = \tilde{U}_i R R' \tilde{V}_a' = Y_{ia}$, providing us with a linear equation on the $\frac{p(p+1)}{2}$ entries in the symmetric $R R'$. For hard-margin matrix factorization, we can therefore recover the entries of $R R'$ by solving a system of linear equations, with a number of variables bounded by the number of observed entries.

**Recovering specific entries** The approach described above requires solving a large system of linear equations (with as many variables as observations). Furthermore, especially when the observations are very sparse (only a small fraction of the entries in the target matrix are observed), the dual solution is much more compact then the prediction matrix: the dual involves a single number for each *observed* entry. It might be desirable to avoid storing the prediction matrix $X^*$ explicitly, and calculate a desired entry $X_{i_0 a_0}^*$, or at least its sign, directly from the dual optimal solution $Q^*$.

Consider adding the constraint $X_{i_0 a_0} > 0$ to the primal SDP (2). If there exists an optimal solution $X^*$ to the original SDP with $X_{i_0 a_0}^* > 0$, then this is also an optimal solution to

the modified SDP, with the same objective value. Otherwise, the optimal solution of the modified SDP is not optimal for the original SDP, and the optimal value of the modified SDP is higher (worse) than the optimal value of the original SDP.

Introducing the constraint $X_{i_0 a_0} > 0$ to the primal SDP (2) corresponds to introducing a new variable $Q_{i_0 a_0}$ to the dual SDP (3), appearing in $Q \otimes Y$ (with $Y_{i_0 a_0} = 1$) but *not* in the objective. In this modified dual, the optimal solution $Q^*$ of the original dual would always be feasible. But, if $X^*_{i_0 a_0} < 0$ in all primal optimal solutions, then the modified primal SDP has a higher value, and so does the dual, and $Q^*$ is no longer optimal for the new dual. By checking the optimality of $Q^*$ for the modified dual, e.g. by attempting to re-optimize it, we can recover the sign of $X^*_{i_0 a_0}$.

We can repeat this test once with $Y_{i_0 a_0} = 1$ and once with $Y_{i_0 a_0} = -1$, corresponding to $X_{i_0 a_0} < 0$. If $Y_{i_0 a_0} X^*_{i_0 a_0} < 0$ (in all optimal solutions), then the dual solution can be improved by introducing $Q_{i_0 a_0}$ with a sign of $Y_{i_0 a_0}$.

**Predictions for new users**  So far, we assumed that learning is done on the known entries in all rows. It is commonly desirable to predict entries in a new partially observed row of $Y$ (a new user), not included in the original training set. This essentially requires solving a "conditional" problem, where $V$ is already known, and a new row of $U$ is learned (the predictor for the new user) based on a new partially observed row of $X$. Using maximum-margin matrix factorization, this is a standard SVM problem.

**Max-norm MMMF as a SDP**  The max-norm variant can also be written as a SDP, with the primal and dual taking the forms:

$$\min t + c \sum_{ia \in S} \xi_{ia} \ \text{s.t.} \ \begin{bmatrix} A & X \\ X' & B \end{bmatrix} \succcurlyeq 0 \quad \begin{matrix} A_{ii}, B_{aa} \leq t \quad \forall i, a \\ y_{ia} X_{ia} \geq 1 - \xi_{ia} \\ \xi_{ia} \geq 0 \end{matrix} \quad \forall ia \in S \quad (5)$$

$$\max \sum_{ia \in S} Q_{ia} \ \text{s.t.} \ \begin{bmatrix} \Gamma & (-Q \otimes Y) \\ (-Q \otimes Y)' & \Delta \end{bmatrix} \succcurlyeq 0 \quad \begin{matrix} \Gamma, \Delta \text{ are diagonal} \\ \text{tr} \, \Gamma + \text{tr} \, \Delta = 1 \\ 0 \leq Q_{ia} \leq c \quad \forall ia \in S \end{matrix} \quad (6)$$

## 5  Generalization Error Bounds for Low Norm Matrix Factorizations

Similarly to standard feature-based prediction approaches, collaborative prediction methods can also be analyzed in terms of their generalization ability: How confidently can we predict entries of $Y$ based on our error on the observed entries $Y_S$? We present here generalization error bounds that holds for *any* target matrix $Y$, and for a random subset of observations $S$, and bound the average error across all entries in terms of the observed margin error[3]. The central assumption, paralleling the i.i.d. source assumption for standard feature-based prediction, is that the observed subset $S$ is picked uniformly at random.

**Theorem 4.** *For all target matrices $Y \in \{\pm 1\}^{n \times m}$ and sample sizes $|S| > n \log n$, and for a uniformly selected sample $S$ of $|S|$ entries in $Y$, with probability at least $1 - \delta$ over*

*the sample selection, the following holds for all matrices $X \in \mathbb{R}^{n \times m}$ and all $\gamma > 0$:*

$$\frac{1}{nm}|\{ia|X_{ia}Y_{ia} \leq 0\}| < \frac{1}{|S|}|\{ia \in S|X_{ia}Y_{ia} \leq \gamma\}|+$$

$$K\frac{\|X\|_{\Sigma}}{\gamma\sqrt{nm}} \sqrt[4]{\ln m} \sqrt{\frac{(n+m)\ln n}{|S|}} + \sqrt{\frac{\ln(1+|\log\|X\|_{\Sigma}/\gamma|)}{|S|}} + \sqrt{\frac{\ln(4/\delta)}{2|S|}} \quad (7)$$

*and*

$$\frac{1}{nm}|\{ia|X_{ia}Y_{ia} \leq 0\}| < \frac{1}{|S|}|\{ia \in S|X_{ia}Y_{ia} \leq \gamma\}|+$$

$$12\frac{\|X\|_{\max}}{\gamma}\sqrt{\frac{n+m}{|S|}} + \sqrt{\frac{\ln(1+|\log\|X\|_{\Sigma}/\gamma|)}{|S|}} + \sqrt{\frac{\ln(4/\delta)}{2|S|}} \quad (8)$$

*Where $K$ is a universal constant that does not depend on $Y$, $n$, $m$, $\gamma$ or any other quantity.*

To understand the scaling of these bounds, consider $n \times m$ matrices $X = UV'$ where the norms of rows of $U$ and $V$ are bounded by $r$, i.e. matrices with $\|X\|_{\max} \leq r^2$. The trace norm of such matrices is bounded by $r^2/\sqrt{nm}$, and so the two bounds agree up to log-factors—the cost of allowing the norm to be low on-average but not uniformly. Recall that the conditional problem, where $V$ is fixed and only $U$ is learned, is a collection of low-norm (large-margin) linear prediction problems. When the norms of rows in $U$ and $V$ are bounded by $r$, a similar generalization error bound on the conditional problem would include the term $\frac{r^2}{\gamma}\sqrt{\frac{n}{|S|}}$, matching the bounds of Theorem 4 up to log-factors—learning *both* $U$ and $V$ does not introduce significantly more error than learning just one of them.

Also of interest is the comparison with bounds for low-rank matrices, for which $\|X\|_{\Sigma} \leq \sqrt{\operatorname{rank} X}\|X\|_{\mathrm{Fro}}$. In particular, for $n \times m$ rank-$k$ $X$ with entries bounded by $B$, $\|X\|_{\Sigma} \leq \sqrt{knm}B$, and the second term in the right-hand side of (7) becomes:

$$K\frac{B}{\gamma}\sqrt[4]{\ln m}\sqrt{\frac{k(n+m)\ln n}{|S|}} \quad (9)$$

Although this is the best (up to log factors) that can be expected from scale-sensitive bounds[4], taking a combinatorial approach, the dependence on the magnitude of the entries in $X$ (and the margin) can be avoided [9].

## 6  Implementation and Experiments

**Ratings**  In many collaborative prediction tasks, the labels are not binary, but rather are discrete "ratings" in several ordered levels (e.g. one star through five stars). Separating $R$ levels by thresholds $-\infty = \theta_0 < \theta_1 < \cdots < \theta_R = \infty$, and generalizing hard-margin constraints for binary labels, one can require $\theta_{Y_{ia}} + 1 \leq X_{ia} \leq \theta_{Y_{ia}+1} - 1$. A soft-margin version of these constraints, with slack variables for the two constraints on each observed rating, corresponds to a generalization of the hinge loss which is a convex bound on the zero/one level-agreement error (ZOE) [10]. To obtain a loss which is a convex bound on the mean-absolute-error (MAE—the difference, in levels, between the predicted level and the true level), we introduce $R - 1$ slack variables for each observed rating—one for each

of the $R-1$ constraints $X_{ia} \geq \theta_r$ for $r < Y_{ia}$ and $X_{ia} \leq \theta_r$ for $r \geq Y_{ia}$. Both of these soft-margin problems ("immediate-threshold" and "all-threshold") can be formulated as SDPs similar to (2)-(3). Furthermore, it is straightforward to learn also the thresholds (they appear as variables in the primal, and correspond to constraints in the dual)—either a single set of thresholds for the entire matrix, or a separate threshold vector for each row of the matrix (each "user"). Doing the latter allows users to "use ratings differently" and alleviates the need to normalize the data.

**Experiments** We conducted preliminary experiments on a subset of the 100K MovieLens Dataset[5], consisting of the 100 users and 100 movies with the most ratings. We used CSDP [11] to solve the resulting SDPs[6]. The ratings are on a discrete scale of one through five, and we experimented with both generalizations of the hinge loss above, allowing per-user thresholds. We compared against WLRA and K-Medians (described in [12]) as "Baseline" learners. We randomly split the data into four sets. For each of the four possible test sets, we used the remaining sets to calculate a 3-fold cross-validation (CV) error for each method (WLRA, K-medians, trace norm and max-norm MMMF with immediate-threshold and all-threshold hinge loss) using a range of parameters (rank for WLRA, number of centers for K-medians, slack cost for MMMF). For each of the four splits, we selected the two MMMF learners with lowest CV ZOE and MAE and the two Baseline learners with lowest CV ZOE and MAE, and measured their error on the held-out test data. Table 1 lists these CV and test errors, and the average test error across all four test sets. On average and on three of the four test sets, MMMF achieves lower MAE than the Baseline learners; on all four of the test sets, MMMF achieves lower ZOE than the Baseline learners.

| Test | | ZOE | | | MAE | |
| Set | Method | CV | Test | Method | CV | Test |
| --- | --- | --- | --- | --- | --- | --- |
| 1 | WLRA rank 2 | 0.547 | 0.575 | K-Medians K=2 | 0.678 | 0.691 |
| 2 | WLRA rank 2 | 0.550 | 0.562 | K-Medians K=2 | 0.686 | **0.681** |
| 3 | WLRA rank 1 | 0.562 | 0.543 | K-Medians K=2 | 0.700 | 0.681 |
| 4 | WLRA rank 2 | 0.557 | 0.553 | K-Medians K=2 | 0.685 | 0.696 |
| Avg. | | | 0.558 | | | 0.687 |
| 1 | max-norm C=0.0012 | 0.543 | **0.562** | max-norm C=0.0012 | 0.669 | **0.677** |
| 2 | trace norm C=0.24 | 0.550 | **0.552** | max-norm C=0.0011 | 0.675 | 0.683 |
| 3 | max-norm C=0.0012 | 0.551 | **0.527** | max-norm C=0.0012 | 0.668 | **0.646** |
| 4 | max-norm C=0.0012 | 0.544 | **0.550** | max-norm C=0.0012 | 0.667 | **0.686** |
| Avg. | | | **0.548** | | | **0.673** |

Table 1: Baseline (top) and MMMF (bottom) methods and parameters that achieved the lowest cross validation error (on the training data) for each train/test split, and the error for this predictor on the test data. All listed MMMF learners use the "all-threshold" objective.

# 7 Discussion

Learning maximum-margin matrix factorizations requires solving a sparse semi-definite program. We experimented with generic SDP solvers, and were able to learn with up to tens of thousands of labels. We propose that just as generic QP solvers do not perform well on SVM problems, special purpose techniques, taking advantage of the very simple structure of the dual (3), are necessary in order to solve large-scale MMMF problems.

SDPs were recently suggested for a related, but different, problem: learning the features

(or equivalently, kernel) that are best for a *single* prediction task [13]. This task is hopeless if the features are completely unconstrained, as they are in our formulation. Lanckriet *et al* suggest constraining the allowed features, e.g. to a linear combination of a few "base feature spaces" (or base kernels), which represent the external information necessary to solve a single prediction problem. It is possible to combine the two approaches, seeking constrained features for multiple related prediction problems, as a way of combining external information (e.g. details of users and of items) and collaborative information.

An alternate method for introducing external information into our formulation is by adding to $U$ and/or $V$ additional fixed (non-learned) columns representing the external features. This method degenerates to standard SVM learning when $Y$ is a vector rather than a matrix.

An important limitation of the approach we have described, is that observed entries are assumed to be uniformly sampled. This is made explicit in the generalization error bounds. Such an assumption is typically unrealistic, as, e.g., users tend to rate items they like. At an extreme, it is often desirable to make predictions based only on positive samples. Even in such situations, it is still possible to learn a low-norm factorization, by using appropriate loss functions, e.g. derived from probabilistic models incorporating the observation process. However, obtaining generalization error bounds in this case is much harder. Simply allowing an arbitrary sampling distribution and calculating the expected loss based on this distribution (which is not possible with the trace norm, but is possible with the max-norm [8]) is not satisfying, as this would guarantee low error on items the user is likely to want anyway, but not on items we predict he would like.

**Acknowledgments**    We would like to thank Sam Roweis for pointing out [7].

## Footnotes

[1]Also known as the *nuclear norm* and the *Ky-Fan n-norm*.

[2] $A \succeq 0$ denotes $A$ is positive semi-definite

[3]The bounds presented here are special cases of bounds for general loss functions that we present and prove elsewhere [8, Section 6.2]. To prove the bounds we bound the Rademacher complexity of bounded trace norm and bounded max-norm matrices (i.e. balls w.r.t. these norms). The unit trace norm ball is the convex hull of outer products of unit norm vectors. It is therefore enough to bound the Rademacher complexity of such outer products, which boils down to analyzing the spectral norm of random matrices. As a consequence of Grothendiek's inequality, the unit max-norm ball is within a factor of two of the convex hull of outer products of sign vectors. The Rademacher complexity of such outer products can be bounded by considering their cardinality.

[4]For general loss functions, bounds as in Theorem 4 depend only on the Lipschitz constant of the loss, and (9) is the best (up to log factors) that can be achieved without explicitly bounding the magnitude of the loss function.

[5]http://www.cs.umn.edu/Research/GroupLens/

[6]Solving with immediate-threshold loss took about 30 minutes on a 3.06GHz Intel Xeon. Solving with all-threshold loss took eight to nine hours. The MATLAB code is available at www.ai.mit.edu/~nati/mmmf

# References

[1] T. Hofmann. Unsupervised learning by probabilistic latent semantic analysis. *Machine Learning Journal*, 42(1):177–196, 2001.

[2] M. Collins, S. Dasgupta, and R. Schapire. A generalization of principal component analysis to the exponential family. In *Advances in Neural Information Processing Systems 14*, 2002.

[3] Nathan Srebro and Tommi Jaakkola. Weighted low rank approximation. In *20th International Conference on Machine Learning*, 2003.

[4] D.D. Lee and H.S. Seung. Learning the parts of objects by non-negative matrix factorization. *Nature*, 401:788–791, 1999.

[5] T. Hofmann. Latent semantic models for collaborative filtering. *ACM Trans. Inf. Syst.*, 22(1):89–115, 2004.

[6] Benjamin Marlin. Modeling user rating profiles for collaborative filtering. In *Advances in Neural Information Processing Systems*, volume 16, 2004.

[7] Maryam Fazel, Haitham Hindi, and Stephen P. Boyd. A rank minimization heuristic with application to minimum order system approximation. In *Proceedings American Control Conference*, volume 6, 2001.

[8] Nathan Srebro. *Learning with Matrix Factorization*. PhD thesis, Massachusetts Institute of Technology, 2004.

[9] N. Srebro, N. Alon, and T. Jaakkola. Generalization error bounds for collaborative prediction with low-rank matrices. In *Advances In Neural Information Processing Systems 17*, 2005.

[10] Amnon Shashua and Anat Levin. Ranking with large margin principle: Two approaches. In *Advances in Neural Information Proceedings Systems*, volume 15, 2003.

[11] B. Borchers. CSDP, a C library for semidefinite programming. *Optimization Methods and Software*, 11(1):613–623, 1999.

[12] B. Marlin. Collaborative filtering: A machine learning perspective. Master's thesis, University of Toronto, 2004.

[13] G. Lanckriet, N. Cristianini, P. Bartlett, L. El Ghaoui, and M. Jordan. Learning the kernel matrix with semidefinite programming. *Journal of Machine Learning Research*, 5:27–72, 2004.
